# Predicting the Risk of Complications in Coronary Artery Bypass Operations using Neural Networks

**Richard P. Lippmann, Linda Kukolich**
MIT Lincoln Laboratory
244 Wood Street
Lexington, MA 02173-0073

**Dr. David Shahian**
Lahey Clinic
Burlington, MA 01805

## Abstract

Experiments demonstrated that sigmoid multilayer perceptron (MLP) networks provide slightly better risk prediction than conventional logistic regression when used to predict the risk of death, stroke, and renal failure on 1257 patients who underwent coronary artery bypass operations at the Lahey Clinic. MLP networks with no hidden layer and networks with one hidden layer were trained using stochastic gradient descent with early stopping. MLP networks and logistic regression used the same input features and were evaluated using bootstrap sampling with 50 replications. ROC areas for predicting mortality using preoperative input features were 70.5% for logistic regression and 76.0% for MLP networks. Regularization provided by early stopping was an important component of improved performance. A simplified approach to generating confidence intervals for MLP risk predictions using an auxiliary "confidence MLP" was developed. The confidence MLP is trained to reproduce confidence intervals that were generated during training using the outputs of 50 MLP networks trained with different bootstrap samples.

## 1 INTRODUCTION

In 1992 there were roughly 300,000 coronary artery bypass operations performed in the United States at a cost of roughly $44,000 per operation. The $13.2 billion total cost of these operations is a significant fraction of health care spending in the United States. This has led to recent interest in comparing the quality of cardiac surgery across hospitals using risk-adjusted procedures and large patient populations. It has also led to interest in better assessing risks for individual patients and in obtaining improved understanding of the patient and procedural characteristics that affect cardiac surgery outcomes.

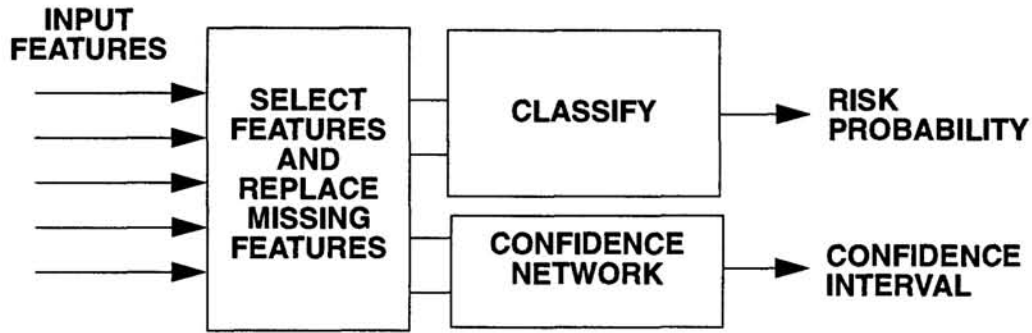

Figure 1. Block diagram of a medical risk predictor.

This paper describes a experiments that explore the use of neural networks to predict the risk of complications in coronary artery bypass graft (CABG) surgery. Previous approaches to risk prediction for bypass surgery used linear or logistic regression or a Bayesian approach which assumes input features used for risk prediction are independent (e.g. Edwards, 1994; Marshall, 1994; Higgins, 1992; O'Conner, 1992). Neural networks have the potential advantages of modeling complex interactions among input features, of allowing both categorical and continuous input features, and of allowing more flexibility in fitting the expected risk than a simple linear or logistic function.

## 2  RISK PREDICTION AND FEATURE SELECTION

A block diagram of the medical risk prediction system used in these experiments is shown in Figure 1. Input features from a patient's medical record are provided as 105 raw inputs, a smaller subset of these features is selected, missing features in this subset are replaced with their most likely values from training data, and a reduced input feature vector is fed to a classifier and to a "confidence network". The classifier provides outputs that estimate the probability or risk of one type of complication. The confidence network provides upper and lower bounds on these risk estimates. Both logistic regression and multilayer sigmoid neural network (MLP) classifiers were evaluated in this study. Logistic regression is the most common approach to risk prediction. It is structurally equivalent to a feed-forward network with linear inputs and one output unit with a sigmoidal nonlinearity. Weights and offsets are estimated using a maximum likelihood criterion and iterative "batch" training. The reference logistic regression classifier used in these experiments was implemented with the S-Plus glm function (Mathsoft, 1993) which uses iteratively reweighted least squares for training and no extra regularization such as weight decay. Multilayer feed-forward neural networks with no hidden nodes (denoted single-layer MLPs) and with one hidden layer and from 1 to 10 hidden nodes were also evaluated as implemented using LNKnet pattern classification software (Lippmann, 1993). An MLP committee classifier containing eight members trained using different initial random weights was also evaluated.

All classifiers were evaluated using a data base of 1257 patients who underwent coronary artery bypass surgery from 1990 to 1994. Classifiers were used to predict mortality, postoperative strokes, and renal failure. Predictions were made after a patient's medical history was obtained (History), after pre-surgical tests had been performed (Post-test), immediately before the operation (Preop), and immediately after the operation (Postop). Bootstrap sampling (Efron, 1993) was used to assess risk prediction accuracy because there were so few

patients with complications in this data base. The number of patients with complications was 33 or 2.6% for mortality, 25 or 2.0% for stroke, and 21 or 1.7% for renal failure. All experiments were performed using 50 bootstrap training sets where a risk prediction technique is trained with a bootstrap training set and evaluated using left-out patterns.

| | $\frac{NComplications}{NHigh}$ | % True Hits |
|---|---|---|
| **HISTORY** | | |
| Age | 27/674 | 4.0% |
| COPD (Chronic Obs. Pul. Disease) | 7/126 | 5.6% |
| **POST-TEST** | | |
| Pulmonary Ventricular Congestion | 8/71 | 11.3% |
| X-ray Cardiomegaly | 6/105 | 5.7% |
| X-ray Pulmonary Edema | 6/21 | 26.6% |
| **PREOP** | | |
| NTG (Nitroglycerin) | 21/447 | 4.7% |
| IABP (Intraaortic Balloon Pump) | 11/115 | 6.6% |
| Urgency Status | 10/127 | 7.9% |
| MI When | 7/64 | 10.9% |
| **POSTOP** | | |
| Blood Used (Packed Cells) | 12/113 | 10.6% |
| Perfusion Time | 9/184 | 4.9% |

Figure 2. Features selected to predict mortality.

The initial set of 105 raw input features included binary (e.g. Male/Female), categorical (e.g. MI When: none, old, recent, evolving), and continuous valued features (e.g. Perfusion Time, Age). There were many missing and irrelevant features and all features were only weakly predictive. Small sets of features were selected for each complication using the following procedures: (1) Select those 10 to 40 features experience and previous studies indicate are related to each complication, (2) Omit features if a univariate contingency table analysis shows the feature is not important, (3) Omit features that are missing for more than 5% of patients, (4) Order features by number of true positives, (5) Omit features that are similar to other features keeping the most predictive, and (7) Add features incrementally as a patient's hospital interaction progresses. This resulted in sets of from 3 to 11 features for the three complications. Figure 2 shows the 11 features selected to predict mortality. The first column lists the features, the second column presents a fraction equal to the number of complications when the feature was "high" divided by the number of times this feature was "high" (A threshold was assigned for continuous and categorical features that provided good separation), and the last column is the second column expressed as a percentage. Classifiers were provided identical sets of input features for all experiments. Continuous inputs to all classifiers were normalized to have zero mean and unit variance, categorical inputs ranged from $-(D-1)/2$ to $(D-1)/2$ in steps of 1.0, where D is the number of categories, and binary inputs were -0.5 or 0.5.

## 3  PERFORMANCE COMPARISONS

Risk prediction was evaluated by plotting and computing the area under receiver operating characteristic (ROC) curves and also by using chi-square tests to determine how accurately classifiers could stratify subjects into three risk categories. Automated experiments were performed using bootstrap sampling to explore the effect of varying the training step size

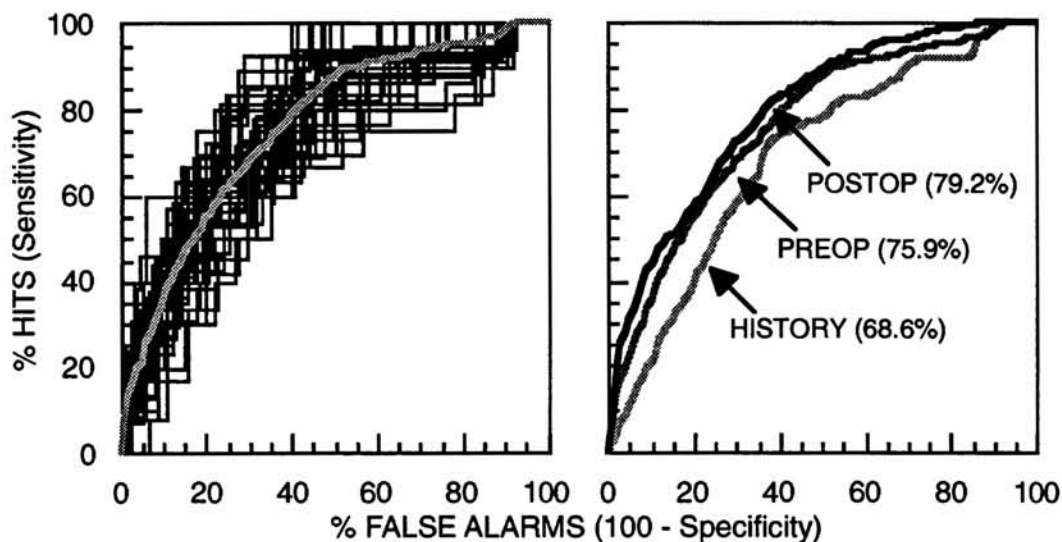

Figure 3. Fifty preoperative bootstrap ROCs predicting mortality using an MLP classifier with two hidden nodes and the average ROC (left), and average ROCS for mortality using history, preoperative, and postoperative features (right).

from 0.005 to 0.1; of using squared-error, cross-entropy, and maximum likelihood cost functions; of varying the number of hidden nodes from 1 to 8; and of stopping training after from 5 to 40 epochs. ROC areas varied little as parameters were varied. Risk stratification, which measures how well classifier outputs approximate posterior probabilities, improved substantially with a cross-entropy cost function (instead of squared error), with a smaller stepsize (0.01 instead of 0.05 or 0.1) and with more training epochs (20 versus 5 or 10). An MLP classifier with two hidden nodes provided good overall performance across complications and patient stages with a cross-entropy cost function, a stepsize of 0.01, momentum of 0.6, and stochastic gradient descent stopping after 20 epochs. A single-layer MLP provided good performance with similar settings, but stopping after 5 epochs. These settings were used for all experiments. The left side of Figure 3 shows the 50 bootstrap ROCs created using these settings for a two-hidden-node MLP when predicting mortality with preoperative features and the ROC created by averaging these curves. There is a large variability in these ROCs due to the small amount of training data. The ROC area varies from 67% to 85% ($\sigma$=4.7) and the sensitivity with 20% false alarms varies from 30% to 79%. Similar variability occurs for other complications. The right side of Figure 3 shows average ROCs for mortality created using this MLP with history, preoperative, and postoperative features. As can be seen, the ROC area and prediction accuracy increases from 68.6% to 79.2% as more input features become available.

Figure 4 shows ROC areas across all complications and patient stages. Only three and two patient stages are shown for stroke and renal failure because no extra features were added at the missing stages for these complications. ROC areas are low for all complications and range from 62% to 80%. ROC areas are highest using postoperative features, lowest using only history features, and increase as more features are added. ROC areas are highest for mortality (68 to 80%) and lower for stroke (62 to 71%) and renal failure (62 to 67%).The MLP classifier with two hidden nodes (MLP) always provided slightly higher ROC areas than logistic regression. The average increase with the MLP classifier was 2.7 percentage

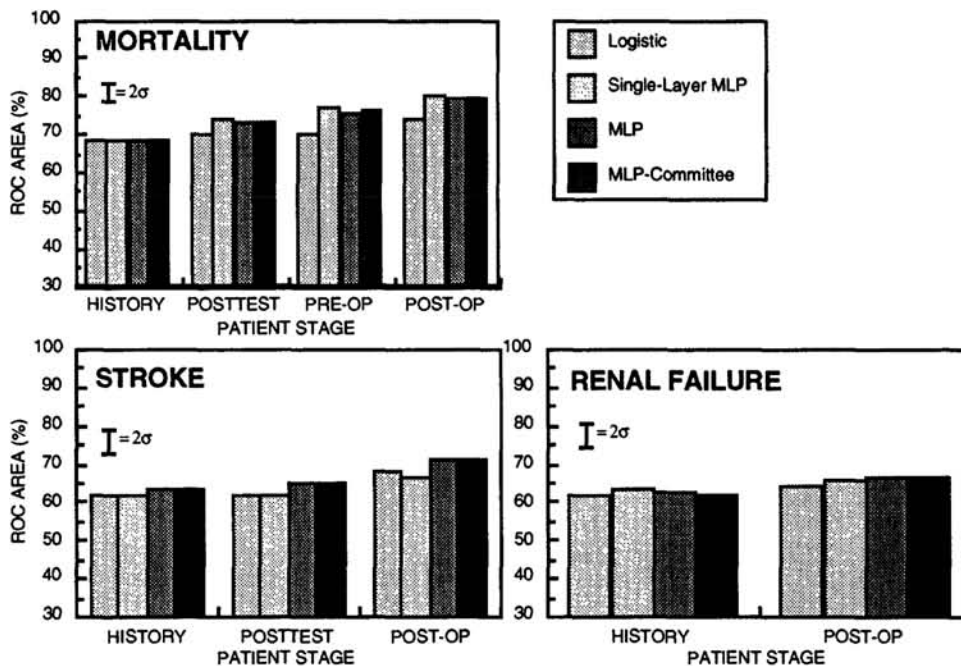

Figure 4. ROC areas across all complications and patient stages for logistic regression, single-layer MLP classifier, two-layer MLP classifier with two hidden nodes, and a committee classifier containing eight two-layer MLP classifiers trained using different random starting weights.

points (the increase ranged from 0.3 to 5.5 points). The single-layer MLP classifier also provided good performance. The average ROC area with the single-layer MLP was only 0.6 percentage points below that of the MLP with two hidden nodes. The committee using eight two-layer MLP classifiers performed no better than an individual two-layer MLP classifier.

Classifier outputs were used to bin or stratify each patient into one of four risk levels (0-5%, 5-10%, and 10-100%) by treating the output as an estimate of the complication posterior probability. Figure 5 shows the accuracy of risk stratification for the MLP classifier for all complications. Each curve was obtained by averaging 50 individual curves obtained using bootstrap sampling as with the ROC curves. Individual curves were obtained by placing each patient into one of the three risk bins based on the MLP output. The x's represent the average MLP output for all patients in each bin. Open squares are the true percentage of patients in each bin who experienced a complication. The bars represent ±2 binomial deviations about the true patient percentages. Risk prediction is accurate if the x's are close to the squares and within the confidence intervals. As can be seen, risk prediction is accurate and close to the actual number of patients who experienced complications. It is difficult, however, to assess risk prediction given the limited numbers of patients in the two highest bins. For example, in Figure 5, the median number of patients with complications was only 2 out of 20 in the middle bin and 2 out of 13 in the upper bin. Good and similar risk stratification, as measured by a chi-square test, was provided by all classifiers. Differences between classifier predictions and true patient percentages were small and not statistically significant.

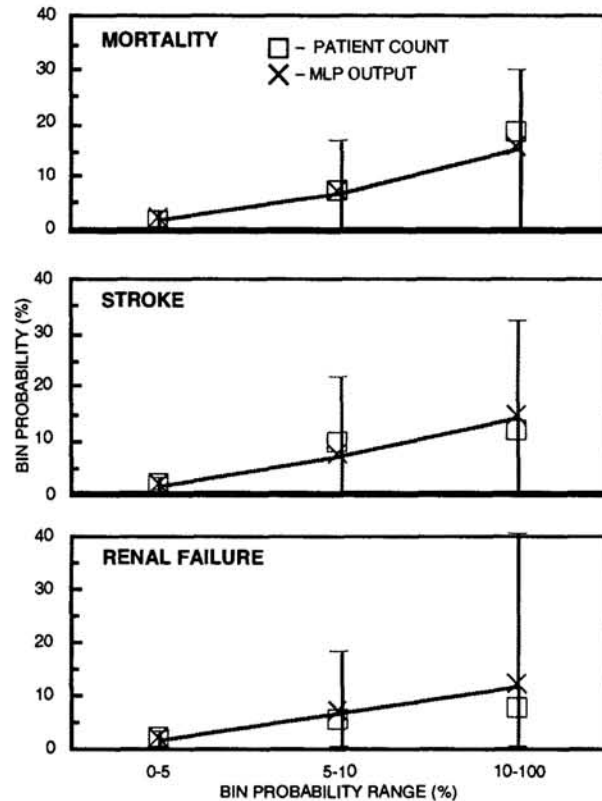

Figure 5. Accuracy of MLP risk stratification for three complications using preoperative features. Open squares are true percentages of patients in each bin with a complication, x's are MLP predictions, bars represent ±2 binomial standard deviation confidence intervals.

## 4 CONFIDENCE MLP NETWORKS

Estimating the confidence in the classification decision produced by a neural network is a critical issue that has received relatively little study. Not being able to provide a confidence measure makes it difficult for physicians and other professionals to accept the use of complex networks. Bootstrap sampling (Efron, 1993) was selected as an approach to generate confidence intervals for medical risk prediction because 1) It can be applied to any type of classifier, 2) It measures variability due to training algorithms, implementation differences, and limited training data, and 3) It is simple to implement and apply. As shown in the top half of Figure 6, 50 bootstrap sets of training data are created from the original training data by resampling with replacement. These bootstrap training sets are used to train 50 bootstrap MLP classifiers using the same architecture and training procedures that were selected for the risk prediction MLP. When a pattern is fed into these classifiers, their outputs provide an estimate of the distribution of the output of the risk prediction MLP. Lower and upper confidence bounds for any input are obtained by sorting these outputs and selecting the 10% and 90% cumulative levels.

It is computationally expensive to have to maintain and query 50 bootstrap MLPs whenever confidence bounds are desired. A simpler approach is to train a single confidence MLP to replicate the confidence bounds predicted by the 50 bootstrap MLPs, as shown in the bot-

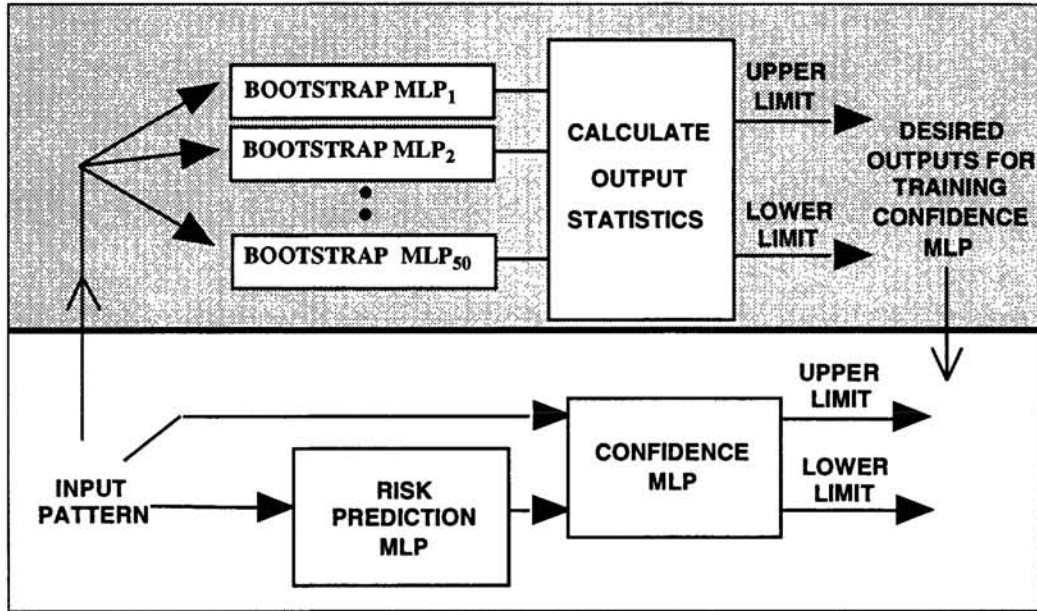

Figure 6.  A confidence MLP trained using 50 bootstrap MLPs produces upper and lower confidence bounds for a risk prediction MLP.

tom half of Figure 6. The the confidence MLP is fed the input pattern and the output of the risk prediction MLP and produces at its output the confidence intervals that would have been produced by 50 bootstrap MLPs. The confidence MLP is a mapping or regression network that replaces the 50 bootstrap networks. It was found that confidence networks with one hidden layer, two hidden nodes, and a linear output could accurately reproduce the upper and lower confidence intervals created by 50 bootstrap two-layer MLP networks. The confidence network outputs were almost always within ±15% of the actual bootstrap bounds. Upper and lower bounds produced by these confidence networks for all patients using preoperative features predicting mortality are show in Figure 7. Bounds are high (±10 percentage points) when the complication risk is near 20% and drop to lower values (±0.4 percentage points) when the risk is near 1%. This relatively simple approach makes it possible to create and replicate confidence intervals for many types of classifiers.

## 5  SUMMARY AND FUTURE PLANS

MLP networks provided slightly better risk prediction than conventional logistic regression when used to predict the risk of death, stroke, and renal failure on 1257 patients who underwent coronary artery bypass operations. Bootstrap sampling was required to compare approaches and regularization provided by early stopping was an important component of improved performance. A simplified approach to generating confidence intervals for MLP risk predictions using an auxiliary "confidence MLP" was also developed. The confidence MLP is trained to reproduce the confidence bounds that were generated during training by 50 MLP networks trained using bootstrap samples. Current research is validating these results using larger data sets, exploring approaches to detect outlier patients who are so different from any training patient that accurate risk prediction is suspect, developing approaches to explaining which input features are important for an individual patient, and determining why MLP networks provide improved performance.

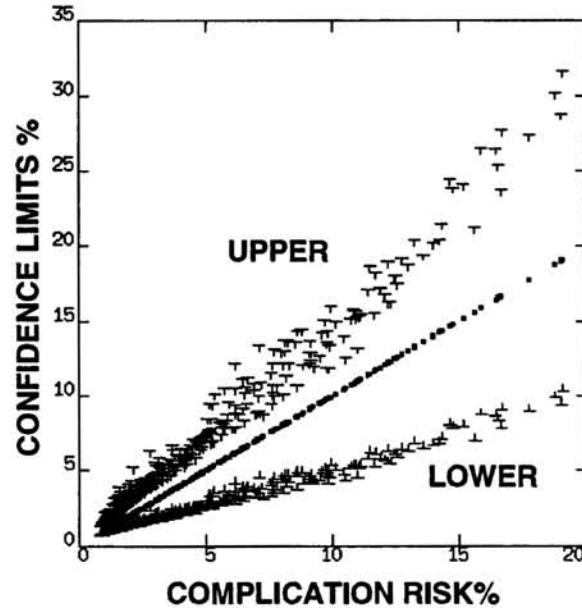

Figure 7. Upper and lower confidence bounds for all patients and preoperative mortality risk predictions calculated using two MLP confidence networks.

## ACKNOWLEDGMENT

This work was sponsored by the Department of the Air Force. The views expressed are those of the authors and do not reflect the official policy or position of the U.S. Government. We wish to thank Stephanie Moisakis and Anne Nilson at the Lahey Clinic and Yuchun Lee at Lincoln Laboratory for assistance in organizing and preprocessing the data.

## BIBLIOGRAPHY

F. Edwards, R. Clark, and M. Schwartz. (1994) Coronary Artery Bypass Grafting: The Society of Thoracic Surgeons National Database Experience. In *Annals Thoracic Surgery*, Vol. 57, 12-19.

Bradley Efron and Robert J. Tibshirani. (1993) An Introduction to the Bootstrap. Monographs on Statistics and Applied Probability 57, New York: Chapman and Hall (1993).

T. Higgins, F. Estafanous, et. al. (1992) Stratification of Morbidity and Mortality Outcome by Preoperative Risk Factors in Coronary Artery Bypass Patients. In Journal of the American Medical Society, Vol. 267, No. 17, 2344-2348.

R. Lippmann, L. Kukolich, and E. Singer. (1993) LNKnet: Neural Network, Machine Learning, and Statistical Software for Pattern Classification. In *Lincoln Laboratory Journal*, Vol. 6, No. 2, 249-268.

Marshall Guillermo, Laurie W. Shroyer, et al. (1994) Bayesian-Logit Model for Risk Assessment in Coronary Artery Bypass Grafting, In *Annals Thoracic Surgery*, Vol. 57, 1492-5000.

G. O'Conner, S. Plume, et. al. (1992) Multivariate Prediction of In-Hospital Mortality Associated with Coronary Artery Bypass Surgery. In *Circulation*, Vol. 85, No. 6, 2110-2118.

Statistical Sciences. (1993) S-PLUS Guide to Statistical and Mathematical Analyses, Version 3.2, Seattle: StatSci, a division of MathSoft, Inc.